# Large Margin Multi-Task Metric Learning

**Shibin Parameswaran**
Department of Electrical and Computer Engineering
University of California, San Diego
La Jolla, CA 92093
sparames@ucsd.edu

**Kilian Q. Weinberger**
Department of Computer Science and Engineering
Washington University in St. Louis
St. Louis, MO 63130
kilian@wustl.edu

## Abstract

Multi-task learning (MTL) improves the prediction performance on multiple, different but related, learning problems through shared parameters or representations. One of the most prominent multi-task learning algorithms is an extension to support vector machines (svm) by Evgeniou et al. [15]. Although very elegant, multi-task svm is inherently restricted by the fact that support vector machines require each *class* to be addressed explicitly with its own weight vector which, in a multi-task setting, requires the different learning tasks to share the same set of classes. This paper proposes an alternative formulation for multi-task learning by extending the recently published large margin nearest neighbor (lmnn) algorithm to the MTL paradigm. Instead of relying on separating hyperplanes, its decision function is based on the nearest neighbor rule which inherently extends to many classes and becomes a natural fit for multi-task learning. We evaluate the resulting multi-task lmnn on real-world insurance data and speech classification problems and show that it consistently outperforms single-task $k$NN under several metrics and state-of-the-art MTL classifiers.

## 1  Introduction

Multi-task learning (MTL) [6, 8, 19] refers to the joint training of multiple problems, enforcing a common intermediate parameterization or representation. If the different problems are sufficiently related, MTL can lead to better generalization and benefit all of the tasks. This phenomenon has been examined further by recent papers which have started to build a theoretical foundation that underpins these initial empirical findings [1, 2, 3].

A well-known application of MTL occurs within the realm of speech recognition. The way different people pronounce the same words differs greatly based on their gender, accent, nationality or other individual characteristics. One can view each possible speaker, or clusters of speakers, as different learning problems that are highly related. Ideally, a speech recognition system should be trained only on data from the user it is intended for. However, annotated data is expensive and difficult to obtain. Therefore, it is highly beneficial to leverage the similarities of data sets from different types of speakers while adapting to the specifics of each particular user [13, 16].

One particularly successful instance of multi-task learning is its adaptation to support vector machines (svm) [14, 15]. Support vector machines are arguably amongst the most successful classification algorithms of all times, however their multi-class extensions such as one-vs-all [4] or clever refinements of the loss functions [10, 21] all require at least one weight vector per class label. As a consequence, the MTL adaptation of svm [15] requires all tasks to share an identical set of labels (or require side-information about task dependencies) for meaningful tranfer of knowledge. This is a serious limitation in many domains (binary or non-binary) where different tasks might not share the same classes (e.g. identifying multiple diseases from a particular patient data).

Recently, Weinberger et al. introduced Large Margin Nearest Neighbor (lmnn) [20], an algorithm that translates the maximum margin learning principle behind svms to $k$-nearest neighbor classification ($k$NN) [9]. Similar to svms, the solution of lmnn is also obtained through a convex optimization problem that maximizes a large margin

between input vectors from different classes. However, instead of positioning a separating hyperplane, lmnn learns a Mahalanobis metric. Weinberger et al. show that the lmnn metric improves the $k$NN classification accuracy to be en par with kernelized svms [20] . One advantage that the $k$NN decision rule has over hyperplane classifiers is its agnosticism towards the number of class labels of a particular data set. A new test point is classified by the majority label of its $k$ closest neighbors within a known training data set — additional classes require no special treatment.

We follow the intuition of Evgeniou et al. [15] and extend lmnn to the multitask setting. Our algorithm learns one metric that is shared amongst all the tasks and one specific metric unique to each task. We show that the combination is still a well-defined pseudo-metric that can be learned in a single convex optimization problem. We demonstrate on several multi-task settings that these shared metrics significantly reduce the overall classification error. Further, our algorithm tends to outperform multi-task neural networks [6] and svm [15] on tasks with many class-labels. To our knowledge, this paper introduces the first multi-task metric learning algorithm for the $k$NN rule that explicitly models the commonalities and specifics of different tasks.

## 2 Large Margin Nearest Neighbor

This section describes the large margin nearest neighbor algorithm as introduced in [20]. For now, we will focus on a single-task learning framework, with a training set consisting of $n$ examples of dimensionality $d$, $\{(\mathbf{x}_i, y_i)\}_{i=1}^n$, where $\mathbf{x}_i \in \mathcal{R}^d$ and $y_i \in \{1, 2, ..., c\}$. Here, $c$ denotes the number of classes. The Mahalanobis distance between two inputs $\mathbf{x}_i$ and $\mathbf{x}_j$ is defined as

$$d_\mathbf{M}(\mathbf{x}_i, \mathbf{x}_j) = \sqrt{(\mathbf{x}_i - \mathbf{x}_j)^\top \mathbf{M}(\mathbf{x}_i - \mathbf{x}_j)}, \quad (1)$$

where $\mathbf{M}$ is a symmetric positive definite matrix ($\mathbf{M} \succeq 0$). The definition in eq. (1) reduces to the Euclidean metric if we set $\mathbf{M}$ to the identity matrix, i.e. $\mathbf{M} = \mathbf{I}$. The lmnn algorithm learns the matrix $\mathbf{M}$ for the Mahalanobis metric[1] in eq. (1) explicitly to enhance $k$-nearest neighbor classification.

Lmnn mimics the non-continuous and non-differentiable leave-one-out classification error of $k$NN with a convex loss function. The loss function encourages the local neighborhood around every input

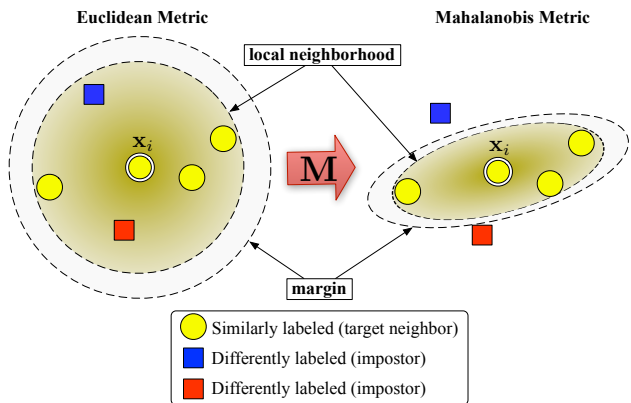

Figure 1: An illustration of a data set before and after lmnn. The circles represent points of equal distance to the vector $\mathbf{x}_i$. The Mahalanobis metric rescales directions to push impostors further away than target neighbors by a large margin.

to stay "pure". Inputs with different labels are pushed away and inputs with a similar label are pulled closer. One of the advantages of lmnn over related work [12, 17] is that the (global) metric is optimized locally, which allows it to work with multi-modal data distributions and encourages better generalization. To achieve this, the algorithm requires $k$ *target neighbors* to be identified for every input prior to learning, which should become the $k$ nearest neighbors after the optimization. Usually, these are picked with the help of side-information, or in the absence thereof, as the $k$ nearest neighbors within the same class based on Euclidean metric. We use the notation $j \rightsquigarrow i$ to indicate that $\mathbf{x}_j$ is a target neighbor of $\mathbf{x}_i$. Lmnn learns a Mahalanobis metric that keeps each input $\mathbf{x}_i$ closer to its *target neighbors* than other inputs with different class labels (*impostors*) — by a large margin. For an input $\mathbf{x}_i$, target neighbor $\mathbf{x}_j$, and impostor $\mathbf{x}_k$, this relation can be expressed as a linear inequality constraint with respect to the squared distance $d_\mathbf{M}^2(\cdot, \cdot)$:

$$d_\mathbf{M}^2(\mathbf{x}_i, \mathbf{x}_k) - d_\mathbf{M}^2(\mathbf{x}_i, \mathbf{x}_j) \geq 1. \quad (2)$$

Eq. (2) is enforced only for the *local* target neighbors. See Fig. 1 for an illustration. Here, all points on the circles have equal distance from $\mathbf{x}_i$. Under the Mahalanobis metric this circle is deformed to an ellipsoid, which causes the impostors (marked as squares) to be further away than the target neighbors.

The semidefinite program (SDP) introduced by [20] moves target neighbors close by minimizing $\sum_{j \rightsquigarrow i} d_\mathbf{M}^2(\mathbf{x}_i, \mathbf{x}_j)$ while penalizing violations of the constraint in eq. (2). The latter is achieved through addi-

tive slack variables $\xi_{ijk} \geq 0$. If we define a set of triples $S = \{(i,j,k) : j \rightsquigarrow i, y_k \neq y_i\}$, the problem can be stated as the SDP shown in Table 1.

This optimization problem has $O(kn^2)$ constraints of type **(1)** and **(2)**, along with the positive semidefinite constraint of a $d \times d$ matrix $\mathbf{M}$. Hence, standard off-the shelf packages are not particularly suited to solve this SDP. For this paper we use the special purpose sub-gradient descent solver, developed in [20], which can handle data sets on the order of tens of thousands of samples. As the optimization problem is not sensitive to the exact choice of the tradeoff constant $\mu$ [20], we set $\mu = 1$ throughout this paper.

$$
\begin{aligned}
&\min_{\mathbf{M}} \sum_{j \rightsquigarrow i} d_{\mathbf{M}}^2(\mathbf{x}_i, \mathbf{x}_j) + \mu \sum_{(i,j,k) \in S} \xi_{ijk} \\
&\textbf{subject to: } (i,j,k) \in S\textbf{:} \\
&\textbf{(1) } d_{\mathbf{M}}^2(\mathbf{x}_i, \mathbf{x}_k) - d_{\mathbf{M}}^2(\mathbf{x}_i, \mathbf{x}_j) \geq 1 - \xi_{ijk} \\
&\textbf{(2) } \xi_{ijk} \geq 0 \\
&\textbf{(3) } \mathbf{M} \succeq 0.
\end{aligned}
$$

Table 1: Convex optimization problem of lmnn.

## 3 Multi-Task learning

In this section, we briefly review the approach presented by Evgeniou et al. [15] that extends svm to multi-task learning (mt-svm). We assume that we are given $T$ *different but related* tasks. Each input $(\mathbf{x}_i, y_i)$ belongs to exactly one of the tasks $1, \ldots, T$, and we let $\mathcal{I}_t$ be the set of indices such that $i \in \mathcal{I}_t$ if and only if the input-label pair $(\mathbf{x}_i, y_i)$ belongs to task $t$. For simplification, throughout this section we will assume a binary classification scenario, in particular $y_i \in \{+1, -1\}$.

Following the original description of [15], mt-svm learns $T$ classifiers $\mathbf{w}_1, \ldots, \mathbf{w}_T$, where each classifier $\mathbf{w}_t$ is specifically dedicated for task $t$. In addition, the authors introduce a global classifier $\mathbf{w}_0$ that captures the commonality among all the tasks. An example $\mathbf{x}_i \in \mathcal{I}_t$ is classified by the rule $\hat{y}_i = \text{sign}(\mathbf{x}_i^\top(\mathbf{w}_0 + \mathbf{w}_t))$. The joint optimization problem is to minimize the following cost:

$$
\min_{\mathbf{w}_0, \ldots, \mathbf{w}_T} \sum_{t=0}^{T} \gamma_t \|\mathbf{w}_t\|_2^2 + \sum_{t=1}^{T} \sum_{i \in I_t} [1 - y_i(\mathbf{w}_0 + \mathbf{w}_t)^\top \mathbf{x}_i]_+ \tag{3}
$$

where $[a]_+ = \max(0, a)$. The constants $\gamma_t \geq 0$ trade-off the regularization of the various tasks. Note that the relative value between $\gamma_0$ and the other $\gamma_{t>0}$ controls the strength of the connection across tasks. In the extreme case, if $\gamma_0 \rightarrow +\infty$, then $\mathbf{w}_0 = \vec{0}$ and all tasks are decoupled; on the other hand, when $\gamma_0$ is small and $\gamma_{t>0} \rightarrow +\infty$ we obtain $\mathbf{w}_{t>0} = \vec{0}$ and all the tasks share the same decision function with weights $\mathbf{w}_0$. Although the mt-svm formulation is very elegant, it requires all tasks to share the same class labels. In the remainder of this paper we will introduce an MTL algorithm based on the $k$NN rule, which does not model each class with its own parameter vector.

## 4 Multi-Task Large Margin Nearest Neighbor

In this section we combine large margin nearest neighbor classification from section 2 with the multi-task learning paradigm from section 3. We follow the MTL setting with $T$ learning tasks. Our goal is to learn a metric $d_t(\cdot, \cdot)$ for each of the $T$ tasks that minimizes the $k$NN leave-one-out classification error. Inspired by the methodology of the previous section, we model the commonalities between various tasks through a shared Mahalanobis metric with $\mathbf{M}_0 \succeq 0$ and the task-specific idiosyncrasies with additional matrices $\mathbf{M}_1, \ldots \mathbf{M}_T \succeq 0$. We define the distance for task $t$ as

$$
d_t(\mathbf{x}_i, \mathbf{x}_j) = \sqrt{(\mathbf{x}_i - \mathbf{x}_j)^\top(\mathbf{M}_0 + \mathbf{M}_t)(\mathbf{x}_i - \mathbf{x}_j)}. \tag{4}
$$

Intuitively, the metric defined by $\mathbf{M}_0$ picks up general trends across multiple data sets and $\mathbf{M}_{t>0}$ specialize the metric further for each particular task. See Fig. 2 for an illustration. If we constrain the matrices $\mathbf{M}_t$ to be positive semi-definite (i.e. $\mathbf{M}_t \succeq 0$), then eq. (4) will result in a well defined pseudo-metric, as we show in section 4.1.

An important aspect of multi-task learning is the appropriate coupling of the multiple learning tasks. We have to ensure that the learning algorithm does not put too much emphasis onto the shared parameters $\mathbf{M}_0$ or the individual

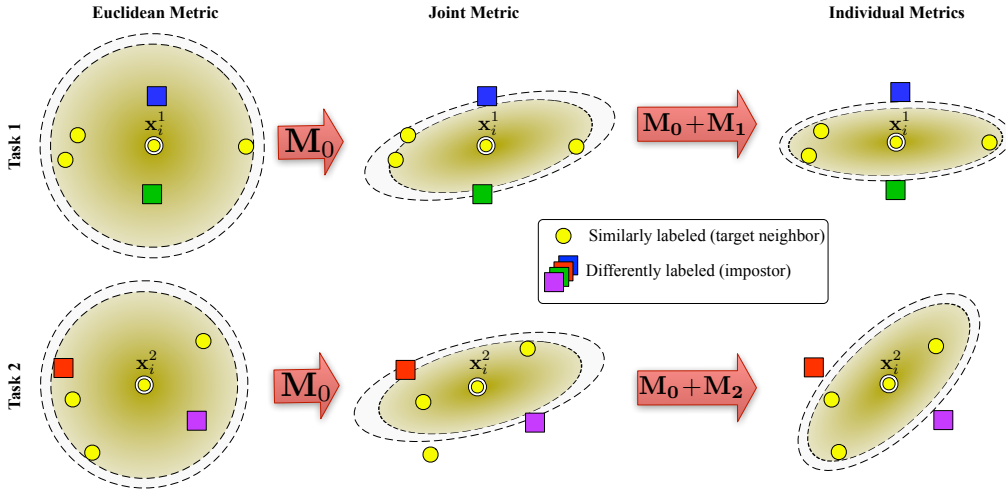

Figure 2: An illustration of mt-lmnn. The matrix $\mathbf{M}_0$ captures the communality between the several tasks, whereas $\mathbf{M}_t$ for $t > 0$ adds the task specific distance transformation.

parameters $\mathbf{M}_1, \ldots, \mathbf{M}_T$. To ensure this balance, we use the regularization term stated below:

$$\min_{\mathbf{M}_0, \ldots, \mathbf{M}_T} \gamma_0 \|\mathbf{M}_0 - \mathbf{I}\|_F^2 + \sum_{t=1}^{T} \gamma_t \|\mathbf{M}_t\|_F^2. \tag{5}$$

The trade-off parameter $\gamma_t$ controls the regularization of $\mathbf{M}_t$ for all $t = 0, 1, \ldots, T$. If $\gamma_0 \to \infty$, the shared metric $\mathbf{M}_0$ reduces to the plain Euclidean metric and if $\gamma_{t>0} \to \infty$, the task-specific metrics $\mathbf{M}_{t>0}$ become irrelevant zero matrices. Therefore, if $\gamma_{t>0} \to \infty$ and $\gamma_0$ is small, we learn a single metric $\mathbf{M}_0$ across all tasks. In this case we want the result to be equivalent to applying lmnn on the union of all data sets. In the other extreme case, when $\gamma_0 = 0$ and $\gamma_{t>0} \to \infty$, we want our formulation to reduce to $T$ independent lmnn algorithms.

Similar to the set of triples $S$ defined in section 2, let $S_t$ be the set of triples restricted to only vectors for task $t$, i.e., $S_t = \{(i, j, k) \in \mathcal{I}_t^3 : j \rightsquigarrow i, y_k \neq y_i\}$. We can combine the regularizer in eq.( 5) with the objective of lmnn applied to each of the $T$ tasks. To ensure well-defined metrics, we add constraints that each matrix is positive semi-definite, i.e. $\mathbf{M}_t \succeq 0$ (see next paragraph for more details). We refer to the resulting algorithm as *multi-task large margin nearest neighbor* (mt-lmnn). The optimization problem is shown in Table 2 and can be solved efficiently after some modifications to the special-purpose solver presented by Weinberger et al. [20]

### 4.1 Theoretical Properties

In this section we verify that our resulting distances are guaranteed to be well-defined pseudo-metrics and that the optimization is convex.

**Theorem 1** *If $M_t \succeq 0$ for all $t = 0, \ldots T$ then the distance functions $d_t(\cdot, \cdot)$, as defined in eq.( 4), are well-defined pseudo-metrics for all $0 \leq t \leq T$.*

The proof of Theorem 1 is completed in two steps: First, as the cone of positive semi-definite matrices is convex, any linear combination of positive semidefinite matrices is also positive semidefinite. This implies that $d_t(\cdot, \cdot)$ is *non-negative*, and it is also trivially *symmetric*. The second part of the proof utilizes the fact that any positive semidefinite matrix $\mathbf{M}$, can be decomposed as $\mathbf{M} = \mathbf{L}^\top \mathbf{L}$, for some matrix $\mathbf{L} \in \mathcal{R}^{d \times d}$. It therefore follows that there exists some matrix $\mathbf{L}_t$ such that $\mathbf{L}_t^\top \mathbf{L}_t = \mathbf{M}_0 + \mathbf{M}_t$. Hence we can rephrase eq.( 4) as

$$d_t(\mathbf{x}_i, \mathbf{x}_j) = \sqrt{(\mathbf{x}_i - \mathbf{x}_j)^\top \mathbf{L}_t^\top \mathbf{L}_t (\mathbf{x}_i - \mathbf{x}_j)}, \tag{6}$$

which is equivalent to the Euclidean distance after the transformation $\mathbf{x}_i \to \mathbf{L}_t \mathbf{x}_i$. It follows that eq.( 6) preserves the *triangular inequality*. This completes the requirements for a *pseudo-metric*. If $\mathbf{L}_t$ is full rank, i.e. $\mathbf{M}_0 + \mathbf{M}_t$ is strictly positive definite, then it also fulfills *identity of indiscernibles*, i.e., $d(\mathbf{x}_i, \mathbf{x}_j) = 0$ if and only if $\mathbf{x}_i = \mathbf{x}_j$ and $d(\cdot, \cdot)$ is a *metric*.

$$\min_{\mathbf{M}_0,\dots,\mathbf{M}_T} \gamma_0 \|\mathbf{M}_0 - \mathbf{I}\|_F^2 + \sum_{t=1}^{T} \left[ \gamma_t \|\mathbf{M}_t\|_F^2 + \sum_{(i,j)\in \mathcal{I}_t, j \rightsquigarrow i} d_t^2(\mathbf{x}_i, \mathbf{x}_j) + \sum_{(i,j,k)\in S_t} \xi_{ijk} \right]$$

**subject to:** $\forall t, \forall (i,j,k) \in S_t$**:**
   **(1)** $d_t^2(\mathbf{x}_i, \mathbf{x}_k) - d_t^2(\mathbf{x}_i, \mathbf{x}_j) \geq 1 - \xi_{ijk}$
   **(2)** $\xi_{ijk} \geq 0$
   **(3)** $\mathbf{M}_0, \mathbf{M}_1, \dots, \mathbf{M}_T \succeq 0$.

Table 2: Convex optimization problem of mt-lmnn.

One of the advantages of lmnn over alternative distance metric learning algorithms, for example NCA [17], is that it can be stated as a convex optimization problem. This allows the global solution to be found efficiently with special purpose solvers [20] or for very large data sets in an online relaxation [7]. It is therefore important to show that our new formulation preserves convexity.

**Theorem 2** *The* mt-lmnn *optimization problem in Table 2 is convex.*

Constraints of type **(2)** and **(3)** are standard linear and positive-semidefinite constraints, which are known to be convex [5]. Convexity remains to be shown for constraints of type **(1)** and the objective. Both access the matrices $\mathbf{M}_t$ exclusively in terms of the squared distance $d^2(\cdot, \cdot)$. This can be expressed as

$$d^2(\mathbf{x}_i, \mathbf{x}_j) = trace(\mathbf{M}_0 \mathbf{v}_{ij} \mathbf{v}_{ij}^\top) + trace(\mathbf{M}_t \mathbf{v}_{ij} \mathbf{v}_{ij}^\top), \tag{7}$$

where $\mathbf{v}_{ij} = (\mathbf{x}_i - \mathbf{x}_j)$. Eq.( 7) is linear in terms of the matrices $\mathbf{M}_t$ and it follows that the constraints of type **(1)** are also linear and therefore trivially convex. Similarly, it follows that all terms in the objective are also linear with the exception of the Frobenius norms in the regularization term. The latter term is quadratic ($\|\mathbf{M}_t\|_F^2 = trace(\mathbf{M}_t^\top \mathbf{M}_t)$) and therefore convex with respect to $\mathbf{M}_t$. The regularization of $\mathbf{M}_0$ can be expanded as $trace(\mathbf{M}_0^\top \mathbf{M}_0 - 2\mathbf{M}_0 + \mathbf{I})$ which has one quadratic and one linear term. The sum of convex functions is convex [5], hence this concludes the proof.

## 5 Results

We evaluate mt-lmnn on the Isolet spoken alphabet recognition[2] and CoIL 2000 dataset[3]. We first provide a brief overview of the two datasets and then present results in various multi-task and domain adaptation settings.

The Isolet dataset was collected from 150 speakers uttering all characters in the English alphabet twice, i.e., each speaker contributed 52 training examples (in total 7797 examples[4]). The task is to classify which letter has been uttered based on several acoustic features – spectral coefficients, contour-, sonorant- and post-sonorant features. The exact feature description can be found in [16]. The speakers are grouped into smaller sets of 30 similar speakers, giving rise to 5 disjoint subsets called isolet1-5. This representation of Isolet lends itself naturally to the multi-task learning regime. We treat each of the subsets as its own classification task ($T = 5$) with $c = 26$ labels. The five tasks differ because the groups of speakers vary greatly in the way they utter the characters of the English alphabets. They are also highly related to each other because all the data is collected from the same utterances (the English alphabets). To remove low variance noise and to speed up computation time we preprocess the Isolet data with PCA [18] and project it onto its leading principal components that capture 95% of the data variance reducing the dimensionality from 617 to 169.

The CoIL dataset contains information of customers of an insurance company. The customer information consists of 86 variables including product usage and socio-demographic data. The training set contains 5822 and the test set 4000 examples. Out of the 86 variables, we used 6 categorical features to create different classification problems, leaving the remaining 80 features as the joint data set. Our target variables consist of attributes 1, 4, 5, 6, 44 and 86,

| Isolet | Euc | U-lmnn | st-lmnn | **mt-lmnn** | st-net | mt-net | st-svm | mt-svm |
|--------|-----|--------|---------|-------------|--------|--------|--------|--------|
| 1 | 13.30% | 6.05% | 5.32% | **3.89**% | 4.74 % | 4.52 % | 8.75% | 5.99% |
| 2 | 18.62% | 6.53% | 5.03% | **3.17**% | 4.62 % | 3.81 % | 9.62% | 5.99% |
| 3 | 21.44% | 8.59% | 10.09% | 6.99% | **6.73** % | 6.92 % | 13.81% | 7.30% |
| 4 | 24.42% | 8.37% | 9.39% | **6.31**% | 7.95 % | 6.51 % | 13.62% | 8.39% |
| 5 | 18.91% | 7.30% | 7.69% | **5.58**% | 5.74 % | 5.61 % | 13.71% | 7.82% |
| Avg | 19.34% | 7.37% | 7.51% | **5.19**% | 5.96 % | 5.48 % | 11.90% | 7.10% |

Table 3: Error rates on **label-compatible** Isolet tasks when tested with *task-specific* train sets.

which indicate customer subtypes, customer age bracket, customer occupation, a discretized percentage of Roman Catholics in that area, contribution from a third party insurance and the last feature is a binary value that signifies if the customer has a caravan insurance policy. The tasks have a different number of output labels but they share the same input data.

Each Isolet subset (task) was divided into randomly selected $60/20/20$ splits of $train/validation/test$ sets. We randomly picked $20\%$ of the CoIL training examples and set them aside for validation purposes. The results were averaged over 10 runs in both cases. The validation subset was used for model selection for mt-lmnn, i.e. choosing the regularization constants $\gamma_t$ and the number of iterations for early stopping. Although our model allows different weights $\gamma_t$ for each task, throughout this paper we only differentiated between $\gamma_0$ and $\gamma = \gamma_{t>0}$. The neighborhood size $k$ was fixed to $k = 3$, which is the setting recommended in the original lmnn publication [20]. For *competing* algorithms, we performed a thorough parameter sweep and reported the best test set results (thereby favoring them over our method).

These two datasets capture the essence of an ideal mt-lmnn application area. Our algorithm is very effective when the feature space is dense and when dealing with multi-label tasks with or without the same set of output labels. This is demonstrated in the first subsection of results. The second subsection provides a brief demonstration of the use of mt-lmnn in the domain adaptation (or cold start) scenario.

## 5.1 Multi-task Learning

We categorized the multi-task learning setting into two different scenarios: label-compatible MTL and label-incompatible MTL. In the label-compatible MTL scenario, all the tasks share the same label set. The label-incompatible scenario arises when applying MTL to a group of multi-class classification tasks that do not share the same set of labels. We demonstrate the applicability and effectiveness of mt-lmnn in both these scenarios in the following sub-sections.

**Label-Compatible Multi-task Learning** The experiments in this setting were conducted on the Isolet data, where isolet1-5 are the 5 tasks and all of them share the same 26 labels.

We compared the performance of our mt-lmnn algorithm with different baselines in table 3. The first 3 algorithms are $k$NN classifiers using different metrics. "Euc" represents the Euclidean metric, "U-lmnn" is the metric obtained from lmnn trained on the union of the training data of all tasks (essentially "pooling" all the data and ignoring the multi-task aspect), "st-lmnn" is single-task lmnn trained independent of other tasks. As additional comparison we have also included results from linear single-task and multi-task support vector machine [15], denoted as "st-svm" and "mt-svm" and non-linear single-task and multi-task neural networks (48 hidden layers) [6] denoted as "st-net" and "mt-net" respectively.

| Isolet | Euc | U-lmnn | st-lmnn | **mt-lmnn** |
|--------|-----|--------|---------|-------------|
| 1 | 9.65% | 4.71% | 5.51% | **4.13**% |
| 2 | 14.01% | 5.19% | 5.29% | **3.94**% |
| 3 | 11.06% | 5.32% | 7.14% | **3.85**% |
| 4 | 12.28% | 5.03% | 7.89% | **4.49**% |
| 5 | 10.67% | 4.17% | 7.11% | **3.65**% |
| Avg | 11.53% | 4.88% | 6.59% | **4.01**% |

Table 4: Error rates when tested with the *union* of train sets from all the tasks.

A special case arises in terms of the $k$NN based classifiers in the label-compatible scenario: during the actual classification step, regardless what metric is used, the $k$NN training data set can either consist of only task specific

| Task | Euc | U-lmnn | st-lmnn | **mt-lmnn** | st-net | mt-net | st-svm |
|------|-----|--------|---------|-------------|--------|--------|--------|
| 1 | 11.25% | 4.27% | 4.48% | 3.44% | 3.92% | **3.43%** | 7.08% |
| 2 | 10.52% | 3.02% | 3.96% | 2.71% | **2.50%** | 2.78% | 6.83% |
| 3 | 14.79% | 6.25% | 6.04% | **5.83%** | 6.67% | 6.39% | 9.58% |
| 4 | 14.79% | 6.25% | 6.46% | **5.52%** | 5.83% | 5.93% | 9.83% |
| 5 | 9.38% | 2.71% | 2.71% | 1.77 % | **1.58%** | 1.67% | 6.17% |
| Avg | 12.15% | 4.50% | 4.73% | **3.85%** | 4.10% | 4.04% | 7.90% |

Table 5: Error rates on Isolet **label-incompatible** tasks with *task-specific* train sets.

| Task | # classes | Euc | st-lmnn | **mt-lmnn** | st-net | mt-net | st-svm |
|------|-----------|-----|---------|-------------|--------|--------|--------|
| 1 | 40 | 24.65% | 13.67% | **12.75%** | 47.45% | 47.05% | 55.68% |
| 2 | 6 | 6.78% | 5.72% | **5.12%** | 17.25% | 19.35% | 36.30% |
| 3 | 10 | 18.48% | 13.28% | **11.06%** | 23.12% | 27.80% | 40.98% |
| 4 | 10 | 7.83 % | 6.05% | **6.00%** | 19.95% | 17.40% | 32.98% |
| 5 | 4 | 33.18 % | 8.23 % | 7.54 % | **3.63%** | **3.63%** | **3.63%** |
| 6 | 2 | 9.25% | 9.12% | 9.10% | **5.95%** | 6.00% | **5.95%** |
| Avg | | 16.70% | 9.35% | **8.60%** | 19.56% | 20.20% | 29.25% |

Table 6: Error rates on CoIL **label-incompatible** tasks. See text for details.

training data or the pooled data from all tasks. The $k$NN results obtained from using pooled training sets at the classification phase is shown in table 4.

Both sets of results, in table 3 and 4, show that mt-lmnn obtains considerable improvement over its single-task counterparts on all 5 tasks and generally outperforms the other multi-task algorithms based on neural networks and support vector machines.

**Label-Incompatible Multi-task Learning** To demonstrate mt-lmnn's ability to learn multiple tasks having different sets of class labels, we ran experiments on the CoIL dataset and on artificially incompatible versions of Isolet tasks. Note that in this setting, mt-svm cannot be used because there is no intuitive way to extend it to label-incompatible multi-class multi-label MTL setting. Also, U-lmnn cannot be used with CoIL data tasks since all of them share the same input.

For each original subset of Isolet we picked 10 labels at random and reduced the dataset to only examples with these labels (resulting in 600 data points per set and different sets of output labels). Table 5 shows the results of the $k$NN algorithm under the various metrics along with single-task and multi-task versions of svm and neural networks on these tasks. Mt-lmnn yields the lowest average performance across all tasks.

The classification error rates on CoIL data tasks are shown in Table 6. The multi-task neural network and svm have a hard time with most of the tasks and, at times perform worse than their single-task versions. Once again, mt-lmnn improves upon its single task counterparts demonstrating the sharing of knowledge between tasks. Both svm and neural networks perform very well on the tasks with the least number of classes, whereas mt-lmnn does very well in tasks with many classes (in particular 40-way classification of task 1).

## 5.2 Domain Adaptation

Domain adaptation attempts to learn a severely undersampled *target* domain, with the help of *source* domains with plenty of data, that may not have the same sample distribution as that of the *target*.

For instance, in the context of speech recognition, one might have a lot of annotated speech recordings from a set of lab volunteers but not much from the client who will use the system. In such cases, we would like the learned classifier to gracefully adapt its recognition / classification rule to the target domain as more data becomes available.

Unlike the previous setting, we now have one specific target task which can be heavily under-sampled. We evaluate the domain adaptation capability of mt-lmnn with isolet1-4 as the *source* and isolet5 as the *target* domain across varying amounts of available labeled target data. The classification errors of $k$NN under the mt-lmnn and U-lmnn metrics are shown in Figure 3.

In the absence of any training data from isolet5 (also referred to as *the cold-start scenario*), we used the global metric $\mathbf{M}_0$ learned by mt-lmnn on tasks isolet1-4. U-lmnn and mt-lmnn global metric perform much better than the Euclidean metric, with U-lmnn giving slightly better classification. With the availability of more data characteristic of the new task, the performance of mt-lmnn improves much faster than U-lmnn. Note that the Euclidean error actually increases with more target data, presumably because utterances from the same speaker might be close together in Euclidean space even if they are from different classes – leading to additional misclassifications.

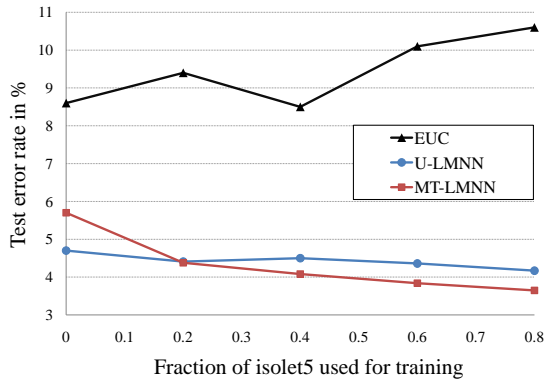

Figure 3: mt-lmnn, U-lmnn and Euclidean test error rates (%) in an unseen task with different sizes of train set.

## 6  Related Work

Caruana was the first to demonstrate results on multi-task learning for k-nearest neighbor regression and locally weighted averaging [6]. The multi-task aspect of their work focused on finding common feature weights across multiple, related tasks. In contrast, our work focuses on classification and learns different metrics with shared components.

Previous work on multi-task learning largely focused on neural networks [6, 8], where a hidden layer is shared between various tasks. This approach is related to our work as it also learns a joint representation across tasks. It differs in the way classification and the optimization are performed. Mt-lmnn uses the $k$NN rule and can be expressed as a convex optimization problem with the accompanying convergence guarantees.

Most recent work in multi-task learning focuses on linear classifiers [11, 15] or kernel machines [14]. Our work was influenced by these publications especially in the way the decoupling of joint and task-specific parameters is performed. However, our method uses a different optimization and learns metrics rather than separating hyperplanes.

## 7  Conclusion

In this paper we introduced a novel multi-task learning algorithm, mt-lmnn. To our knowledge, it is the first metric learning algorithm that embraces the multi-task learning paradigm that goes beyond feature re-weighting for pooled training data. We demonstrated the abilities of mt-lmnn on real-world datasets. Mt-lmnn consistently outperformed single-task metrics for $k$NN in almost all of the learning settings and obtains better classification results than multi-task neural networks and support-vector machines. Addressing a major limitation of mt-svm, mt-lmnn is applicable (and effective) on multiple *multi-class* tasks with *different* sets of classes.

This MTL framework can also be easily adapted for other metric learning algorithms including the online version of lmnn [7]. A further research extension is to incorporate known structure by introducing additional *sub-global* metrics that are shared only by a strict subset of the tasks.

The nearest neighbor classification rule is a natural fit for multi-task learning, if accompanied with a suitable metric. By extending one of the state-of-the-art metric learning algorithms to the multi-task learning paradigm, mt-lmnn provides a more integrative methodology for metric learning across multiple learning problems.

### Acknowledgments

The authors would like to thank Lawrence Saul for helpful discussions. This research was supported in part by the UCSD FWGrid Project, NSF Research Infrastructure Grant Number EIA-0303622.

## Footnotes

[1]For simplicity we will refer to *pseudo-metrics* also as *metrics* as the distinction has no implications for our algorithm.

[2]Available for download from the UCI Machine Learning Repository.

[3]Available for download at http://kdd.ics.uci.edu/databases/tic/tic.html

[4]Three examples are historically missing.

# References

[1] B. Bakker and T. Heskes. Task clustering and gating for bayesian multitask learning. *Journal of Machine Learning Research*, 4:83–99, 2003.

[2] S. Ben-David, J. Gehrke, and R. Schuller. A theoretical framework for learning from a pool of disparate data sources. In *KDD*, pages 443–449, 2002.

[3] S. Ben-David and R. Schuller. Exploiting task relatedness for mulitple task learning. In *COLT*, pages 567–580, 2003.

[4] B. Boser, I. Guyon, and V. Vapnik. A training algorithm for optimal margin classifiers. In *Proceedings of the fifth annual workshop on Computational learning theory*, pages 144–152. ACM New York, NY, USA, 1992.

[5] S. Boyd and L. Vandenberghe. *Convex Optimization*. Cambridge University Press, 2004.

[6] R. Caruana. Multitask learning. *Machine Learning*, 28(1):41–75, 1997.

[7] G. Chechik, U. Shalit, V. Sharma, and S. Bengio. An online algorithm for large scale image similarity learning. In Y. Bengio, D. Schuurmans, J. Lafferty, C. K. I. Williams, and A. Culotta, editors, *Advances in Neural Information Processing Systems 22*, pages 306–314. 2009.

[8] R. Collobert and J. Weston. A unified architecture for NLP: Deep neural networks with multitask learning. In *Proceedings of the 25th international conference on Machine learning*, pages 160–167. ACM New York, NY, USA, 2008.

[9] T. Cover and P. Hart. Nearest neighbor pattern classification. In *IEEE Transactions in Information Theory, IT-13*, pages 21–27, 1967.

[10] K. Crammer and Y. Singer. On the algorithmic implementation of multiclass kernel-based vector machines. *The Journal of Machine Learning Research*, 2:265–292, 2002.

[11] H. Daumé. Frustratingly easy domain adaptation. In *Annual Meeting-Association for Computational Linguistics*, volume 45, page 256, 2007.

[12] J. Davis, B. Kulis, P. Jain, S. Sra, and I. Dhillon. Information-theoretic metric learning. *Proceedings of the 24th international conference on Machine learning*, 2007.

[13] V. Digalakis, D. Rtischev, and L. Neumeyer. Fast speaker adaptation using constrained estimation of Gaussian mixtures. *IEEE Trans. on Speech and Audio Processing*, pages 357–366, 1995.

[14] T. Evgeniou, C. Micchelli, and M. Pontil. Learning multiple tasks with kernel methods. *Journal of Machine Learning Research*, 6(1):615, 2006.

[15] T. Evgeniou and M. Pontil. Regularized multi–task learning. In *KDD*, pages 109–117, 2004.

[16] M. A. Fanty and R. Cole. Spoken letter recognition. In *Advances in Neural Information Processing Systems 4*, page 220. MIT Press, 1990.

[17] J. Goldberger, S. Roweis, G. Hinton, and R. Salakhutdinov. Neighbourhood components analysis. In L. K. Saul, Y. Weiss, and L. Bottou, editors, *Advances in Neural Information Processing Systems 17*, pages 513–520, Cambridge, MA, 2005. MIT Press.

[18] I. T. Jolliffe. *Principal Component Analysis*. Springer-Verlag, New York, 1986.

[19] A. Quattoni, C. X., C. M., and D. T. A projected subgradient method for scalable multi-task learning. *Massachusetts Institute of Technology, Technical Report*, 2008.

[20] K. Q. Weinberger and L. K. Saul. Distance metric learning for large margin nearest neighbor classification. *The Journal of Machine Learning Research*, 10:207–244, 2009.

[21] J. Weston and C. Watkins. Support vector machines for multi-class pattern recognition. In *ESANN*, page 219, 1999.

